# Variational Linear Response

**Manfred Opper**[1]                    **Ole Winther**[2]

[1] Neural Computing Research Group, School of Engineering and Applied Science,
Aston University, Birmingham B4 7ET, United Kingdom
[2] Informatics and Mathematical Modelling, Technical University of Denmark,
R. Petersens Plads, Building 321, DK-2800 Lyngby, Denmark
opperm@aston.ac.uk owi@imm.dtu.dk

## Abstract

A general linear response method for deriving improved estimates of cor-
relations in the variational Bayes framework is presented. Three applica-
tions are given and it is discussed how to use linear response as a general
principle for improving mean field approximations.

## 1 Introduction

Variational and related mean field techniques have attracted much interest as methods for
performing approximate Bayesian inference, see e.g. [1]. The maturity of the field has
recently been underpinned by the appearance of the variational Bayes method [2, 3, 4] and
associated software making it possible with a window based interface to define and make
inference for a diverse range of graphical models [5, 6].

Variational mean field methods have shortcomings as thoroughly discussed by Mackay [7].
The most important is that it based upon the variational assumption of independent vari-
ables. In many cases where the effective coupling between the variables are weak this
assumption works very well. However, if this is not the case, the variational method can
grossly underestimate the width of marginal distributions because variance contributions
induced by other variables are ignored as a consequence of the assumed independence.
Secondly, the variational approximation may be non-convex which is indicated by the oc-
currence of multiple solutions for the variational distribution. This is a consequence of
the fact that a possibly complicated multi-modal distribution is approximated by a simpler
uni-modal distribution.

Linear response (LR) is a perturbation technique that gives an improved estimate of the cor-
relations between the stochastic variables by expanding around the solution to variational
distribution [8]. This means that we can get non-trivial estimates of correlations from the
factorizing variational distribution. In many machine learning models, e.g. Boltzmann ma-
chine learning [9] or probabilistic Independent Component Analysis [3, 10], the M-step of
the EM algorithm depend upon the covariances of the variables and LR has been applied
with success in these cases [9, 10].

Variational calculus is in this paper used to derive a general linear response correction from
the variational distribution. It is demonstrated that the variational LR correction can be
calculated as systematically the variational distribution in the Variational Bayes framework

(albeit at a somewhat higher computational cost). Three applications are given: a model with a quadratic interactions, a Bayesian model for estimation of mean and variance of a 1D Gaussian and a Variational Bayes mixture of multinomials (i.e. for modeling of histogram data). For the two analytically tractable models (the Gaussian and example two above), it is shown that LR gives the correct analytical result where the variational method does not. The need for structured approximations, see e.g. [5] and references therein, that is performing exact inference for solvable subgraphs, might thus be eliminated by the use of linear response.

We define a general probabilistic model $\mathcal{M}$ for data $\mathbf{y}$ and model parameters $\mathbf{s}$: $p(\mathbf{s}, \mathbf{y}) = p(\mathbf{s}, \mathbf{y}|\mathcal{M})$. The objective of a Bayesian analysis are typically the following: to derive the marginal likelihood $p(\mathbf{y}|\mathcal{M}) = \int d\mathbf{s}\, p(\mathbf{s}, \mathbf{y}|\mathcal{M})$ and marginal distributions e.g. the one-variable $p_i(s_i|\mathbf{y}) = \frac{1}{p(\mathbf{y})} \int \prod_{k \neq i} ds_k p(\mathbf{s}, \mathbf{y})$ and the two-variable $p_{ij}(s_i, s_j|\mathbf{y}) = \frac{1}{p(\mathbf{y})} \int \prod_{k \neq i,j} ds_k p(\mathbf{s}, \mathbf{y})$. In this paper, we will only discuss how to derive linear response approximations to marginal distributions. Linear response corrected marginal likelihoods can also be calculated, see Ref. [11].

The paper is organized as follows: in section 2 we discuss how to use the marginal likelihood as a generating functions for deriving marginal distributions. In section 4 we use this result to derive the linear response approximation to the two-variable marginals and derive an explict solution of these equations in section 5. In section 6 we discuss why LR in the cases where the variational method gives a reasonable solution will give an even better result. In section 7, we give the three applications and in section we conclude and discuss how to combine the mean field approximation (variational, Bethe,...) with linear response to give more precise mean field approaches.

After finishing this paper we have become aware of the work of Welling and Teh [12, 13] which also contains the result eq. (8) and furthermore extend linear response to the Bethe approximation, give several general results for the properties of linear response estimates and derive belief propagation algorithms for computing the linear response estimates. The new contributions of this paper compared to Refs. [12, 13] are the explicit solution of the linear response equations, the discussion of the expected increased quality of linear response estimates, the applications of linear response to concrete examples especially in relation to variational Bayes and the discussion of linear response and mean field methods beyond variational.

## 2 Generating Marginal Distributions

In this section it is shown how exact marginal distributions can be obtained from functional derivatives of a generating function (the log partition function). In the derivation of the variational linear response approximation to the two-variable marginal distribution $p_{ij}(s_i, s_j|\mathbf{y})$, we can use result by replacing the exact marginal distribution with the variational approximation. To get marginal distributions we introduce a generating function

$$Z[\mathbf{a}] = \int d\mathbf{s}\, p(\mathbf{s}, \mathbf{y}) e^{\sum_i a_i(s_i)} \tag{1}$$

which is a functional of the arbitrary functions $a_i(s_i)$ and $\mathbf{a}$ is shorthand for the vector of functions $\mathbf{a} = (a_1(s_1), a_2(s_2), \ldots, a_N(s_N))$. We can now obtain the marginal distribution $p(s_i|\mathbf{y}, \mathbf{a})$ by taking the functional derivative[1] with respect to $a_i(s_i)$:

$$\frac{\delta}{\delta a_i(s_i)} \ln Z[\mathbf{a}] = \frac{e^{a_i(s_i)}}{Z[\mathbf{a}]} \int \prod_{k \neq i} \left\{ d\hat{s}_k e^{a_k(\hat{s}_k)} \right\} p(\hat{\mathbf{s}}, \mathbf{y}) = p_i(s_i|\mathbf{y}, \mathbf{a}) \,. \tag{2}$$

Setting $\mathbf{a} = 0$ above gives the promised result. The next step is to take the second derivative. This will give us a function that are closely related to the two-variable marginal distribution. A careful derivation gives

$$
\begin{aligned}
B_{ij}(s_i, s'_j) &\equiv \left. \frac{\delta^2 \ln Z[\mathbf{a}]}{\delta a_j(s'_j) \delta a_i(s_i)} \right|_{\mathbf{a}=0} = \left. \frac{\delta p_i(s_i|\mathbf{y}, \mathbf{a})}{\delta a_j(s'_j)} \right|_{\mathbf{a}=0} \\
&= \delta_{ij} \delta(s_i - s'_j) p_i(s_i|\mathbf{y}) + (1 - \delta_{ij}) p_{ij}(s_i, s'_j|\mathbf{y}) - p_i(s_i|\mathbf{y}) p_j(s'_j|\mathbf{y})
\end{aligned}
\tag{3}
$$

Performing an average of $s_i^m (s'_j)^n$ over $B_{ij}(s_i, s'_j)$, it is easy to see that $B_{ij}(s_i, s'_j)$ gives the 'mean-subtracted' marginal distributions. In the two next sections, variational approximations to the single variable and two-variable marginals are derived.

## 3   Variational Learning

In many models of interest, e.g. mixture models, exact inference scales exponentially with the size of the system. It is therefore of interest to come up with polynomial approximations. A prominent method is the variational, where a simpler factorized distribution $q(\mathbf{s}) = \prod_i q_i(s_i)$ is used instead of the posterior distribution. Approximations to the marginal distributions $p_i(s_i|\mathbf{y})$ and $p_{ij}(s_i, s_j|\mathbf{y})$ are now simply $q_i(s_i)$ and $q_i(s_i)q_j(s_j)$. The purpose of this paper is to show that it is possible within the variational framework to go beyond the factorized distribution for two-variable marginals. For this purpose we need the distribution $q(\mathbf{s})$ which minimizes the KL-divergence or 'distance' between $q(\mathbf{s})$ and $p(\mathbf{s}|\mathbf{y})$:

$$
KL(q(\mathbf{s})||p(\mathbf{s}|\mathbf{y})) = \int d\mathbf{s}\, q(\mathbf{s}) \ln \frac{q(\mathbf{s})}{p(\mathbf{s}|\mathbf{y})} \ .
\tag{4}
$$

The variational approximation to the Likelihood is obtained from

$$
-\ln Z_v[\mathbf{a}] = \int d\mathbf{s}\, q(\mathbf{s}) \ln \frac{q(\mathbf{s})}{p(\mathbf{s}, \mathbf{y}) e^{\sum_k a_k(\hat{s}_k)}} = -\ln Z[\mathbf{a}] + KL(q(\mathbf{s})||p(\mathbf{s}|\mathbf{y}, \mathbf{a})) \ ,
$$

where $\mathbf{a}$ has been introduced to be able use $q_i(s_i|\mathbf{a})$ as a generating function. Introducing Lagrange multipliers $\{\lambda_i\}$ as enforce normalization and minimizing $KL + \sum_i \lambda_i (\int ds_i q_i(s_i) - 1)$ with respect to $q_i(s_i)$ and $\lambda_i$, one finds

$$
q_i(s_i|\mathbf{a}) = \frac{e^{a_i(s_i) + \int \prod_{k \neq i} \{ds_k q_k(s_k|\mathbf{a})\} \ln p(\mathbf{s}, \mathbf{y})}}{\int d\hat{s}_i e^{a_i(\hat{s}_i) + \int \prod_{k \neq i} \{d\hat{s}_k q_k(\hat{s}_k|\mathbf{a})\} \ln p(\hat{\mathbf{s}}, \mathbf{y})}} \ .
\tag{5}
$$

Note that $q_i(s_i|\mathbf{a})$ depends upon all $\mathbf{a}$ through the implicit dependence in the $q_k$s appearing on the right hand side. Writing the posterior in terms of 'interaction potentials', i.e. as a factor graph

$$
p(\mathbf{s}, \mathbf{y}) = \prod_i \psi_i(s_i) \prod_{i>j} \psi_{i,j}(s_i, s_j) \dots \ ,
\tag{6}
$$

it is easy to see that potentials that do not depend upon $s_i$ will drop out of variational distribution. A similar property will be used below to simplify the variational two-variable marginals.

## 4   Variational Linear Response

Eq. (3) shows that we can obtain the two-variable marginal as the derivative of the marginal distribution. To get the variational linear response approximation we exchange the exact marginal with the variational approximation eq. (5) in eq. (3). In section 6 an argument is

given for why one can expect the variational approach to work in many cases and why the linear response approximation gives improved estimates of correlations in these cases.

Defining the variational 'mean subtracted' two-variable marginal as

$$C_{ij}(s_i, s'_j|\mathbf{a}) \equiv \frac{\delta q_i(s_i|\mathbf{a})}{\delta a_j(s'_j)} , \tag{7}$$

it is now possible to derive an expression corresponding to eq. (3). What makes the derivation a bit cumbersome is that it necessary to take into account the implicit dependence of $a_j(s'_j)$ in $q_k(s_k|\mathbf{a})$ and the result will consequently be expressed as a set of linear integral equations in $C_{ij}(s_i, s'_j|\mathbf{a})$. These equations can be solved explicitly, see section 5 or can as suggested by Welling and Teh [12, 13] be solved by belief propagation.

Taking into account both explicit and implicit $\mathbf{a}$ dependence we get the variational linear response theorem:

$$
\begin{aligned}
C_{ij}(s_i, s'_j|\mathbf{a}) \;\; = \;\; & \delta_{ij}\left\{\delta(s_i - s'_j)q_i(s_i|\mathbf{a}) - q_i(s_i|\mathbf{a})q_j(s'_j|\mathbf{a})\right\} \\
& + q_i(s_i|\mathbf{a})\sum_{l\neq i}\int\prod_{k\neq i}ds_k\prod_{k\neq i,l}q_k(s_k|\mathbf{a})C_{lj}(s_l, s'_j|\mathbf{a}) \\
& \qquad\qquad\qquad \times\left\{\ln p(\mathbf{s}, \mathbf{y}) - \int ds_i q_i(s_i|\mathbf{a})\ln p(\mathbf{s}, \mathbf{y})\right\} .
\end{aligned}
\tag{8}
$$

The first term represents the normal variational correlation estimate and the second term is linear response correction which expresses the coupling between the two-variable marginals.

Using the factorization of the posterior eq. (6), it is easily seen that potentials that do not depend on both $s_i$ and $s_l$ will drop out in the last term. This property will make the calculations for the most variational Bayes models quite simple since this means that one only has to sum over variables that are directly connected in the graphical model.

## 5  Explicit Solution

The integral equation can be simplified by introducing the symmetric kernel

$$K_{ij}(s, s') = (1 - \delta_{ij})\left(\langle\ln p(\mathbf{s}, \mathbf{y})\rangle_{\backslash(i,j)} - \langle\ln p(\mathbf{s}, \mathbf{y})\rangle_{\backslash j} - \langle\ln p(\mathbf{s}, \mathbf{y})\rangle_{\backslash i} + \langle\ln p(\mathbf{s}, \mathbf{y})\rangle\right) ,$$

where the brackets $\langle\ldots\rangle_{\backslash(i,j)} = \langle\ldots\rangle_{q\backslash(i,j)}$ denote expectations over $q$ for all variables, except for $s_i$ and $s_j$ and similarly for $\langle\ldots\rangle_{\backslash i}$. One can easily show that $\int ds\, q_i(s)\, K_{ij}(s, s') = 0$. Writing $C$ in the form

$$C_{ij}(s, s') = q_i(s)q_j(s')\left\{\delta_{ij}\left(\frac{\delta(s - s')}{q_j(s')} - 1\right) + R_{ij}(s, s')\right\} , \tag{9}$$

we obtain an integral equation for the function $R$

$$R_{ij}(s, s') = \sum_l\int d\tilde{s}\, q_l(\tilde{s})K_{il}(s, \tilde{s})R_{lj}(\tilde{s}, s') + K_{ij}(s, s') . \tag{10}$$

This result can most easily be obtained by plugging the definition eq. (9) into eq. (8) and using that $\int ds\, q_i(s)\, R_{ij}(s, s') = 0$. For many applications, kernels can be written in the form of sums of pairwise multiplicative 'interactions', i.e.

$$K_{ij}(s, s') = \sum_{\alpha\alpha'} J_{ij}^{\alpha\alpha'}\phi_i^\alpha(s)\phi_j^{\alpha'}(s') \tag{11}$$

with $\langle \phi_i^\alpha \rangle_q = 0$ for all $i$ and $\alpha$ then the solution will be on the form $R_{ij}(s,s') = \sum_{\alpha\alpha'} A_{ij}^{\alpha\alpha'} \phi_i^\alpha(s)\phi_j^{\alpha'}(s')$. The integral equation reduces to a system of linear equations for the coefficients $A_{ij}^{\alpha\alpha'}$.

We now discuss the simplest case where $K_{ij}(s,s') = J_{ij}\phi_i(s)\phi_j(s')$. This is obtained if the model has only pairwise interactions of the quadratic form $\psi_{ij}(s,s') = e^{J_{ij}\Phi_i(s)\Phi_j(s')}$, where $\phi_i(s) = \Phi_i(s) - \langle \Phi_i \rangle_q$. Using $R_{ij}(s,s') = A_{ij}\phi_i(s)\phi_j(s')$ and augmenting the matrix of $J_{ij}$'s with the diagonal elements $J_{ii} \equiv -\frac{1}{\langle \phi_i^2 \rangle_q}$ yield the solution

$$A_{ij} = -J_{ii}J_{jj}\left( D(J_{ii}) - \mathbf{J}^{-1} \right)_{ij} , \tag{12}$$

where $D(J_{ii})$ is a diagonal matrix with entries $J_{ii}$. Using (9), this result immediately leads to the expression for the correlations

$$\langle \phi_i \phi_j \rangle = \langle \Phi_i \Phi_j \rangle - \langle \Phi_i \rangle \langle \Phi_j \rangle = -(\mathbf{J}^{-1})_{ij} . \tag{13}$$

## 6 Why Linear Response Works

It may seem paradoxical that an approximation which is based on uncorrelated variables allows us to obtain a nontrivial result for the neglected correlations. To shed more light on this phenomenon, we would like to see how the true partition function, which serves as a generating function for expectations, differs from the mean field one when the approximating mean field distribution $q$ are close. We will introduce into the generating function eq. (1) the parameter $\epsilon$:

$$Z_\epsilon[\mathbf{a}] = \int d\mathbf{s}\, q(\mathbf{s}) e^{\epsilon(\sum_i a_i(s_i) + \ln p(\mathbf{s}|\mathbf{y}) - \ln q(\mathbf{s}))} \tag{14}$$

which serves as a bookkeeping device for collecting relevant terms, when $\ln p(\mathbf{s}|\mathbf{y}) - \ln q(\mathbf{s})$ is assumed to be small. At the end we will set $\epsilon = 1$ since $Z[\mathbf{a}] = Z_{\epsilon=1}[\mathbf{a}]$. Then expanding the partition function to first order in $\epsilon$, we get

$$
\begin{aligned}
\ln Z_\epsilon[\mathbf{a}] &= \epsilon\left( \sum_i \langle a_i(s_i) \rangle_q + \langle \ln p(\mathbf{s}|\mathbf{y}) - \ln q(\mathbf{s}) \rangle_q \right) + O(\epsilon^2) \\
&= \epsilon\left( \sum_i \langle a_i(s_i) \rangle_q - KL(q\|p) \right) + O(\epsilon^2) .
\end{aligned}
\tag{15}
$$

Keeping only the linear term, setting $\epsilon = 1$ and inserting the minimizing mean field distribution for $q$ yields

$$p_i(s|\mathbf{y},\mathbf{a}) = \frac{\delta \ln Z}{\delta a_i(s)} = q_i(s|\mathbf{a}) + \mathcal{O}(\epsilon^2) . \tag{16}$$

Hence the computation of the correlations via

$$B_{ij}(s,s') = \frac{\delta^2 \ln Z}{\delta a_i(s)\delta a_j(s')} = \frac{\delta p_i(s|\mathbf{a})}{\delta a_j(s')} = \frac{\delta q_i(s|\mathbf{a})}{\delta a_j(s')} + \mathcal{O}(\epsilon^2) = C_{ij}(s,s') + \mathcal{O}(\epsilon^2) \tag{17}$$

can be assumed to incorporate correctly effects of *linear order* in $\ln p(\mathbf{s}|\mathbf{a}) - \ln q(\mathbf{s})$. On the other hand, one should expect $p(s_i, s_j|\mathbf{y}) - q_i(s_i)q_j(s_j)$ to be order $\epsilon$. Although the above does not prove that diagonal correlations are estimated more precisely from $C_{ii}(s,s')$ than from $q_i(s)$–only that both are correct to linear order in $\epsilon$—one often observes this in practice, see below.

## 7 Applications

### 7.1 Quadratic Interactions

The quadratic interaction model—$\ln \psi_{ij}(s_i, s_j) = s_i J_{ij} s_j$ and arbitrary $\psi(s_i)$, i.e. $\ln p(\mathbf{s}, \mathbf{y}) = \sum_i \ln \psi_i(s_i) + \frac{1}{2} \sum_{i \neq j} s_i J_{ij} s_j + \text{constant}$—is used in many contexts, e.g. the Boltzmann machine, independent component analysis and the Gaussian process prior. For this model we can immediately apply the result eq. (13) to get

$$\langle s_i s_j \rangle - \langle s_i \rangle \langle s_j \rangle = -(\mathbf{J}^{-1})_{ij} \tag{18}$$

where we have set $J_{ii} = -1/(\langle s_i^2 \rangle_q - \langle s_i \rangle_q^2)$.

We can apply this to the Gaussian model $\ln \psi_i(s_i) = h_i s_i + A_i s_i^2/2$, The variational distribution is Gaussian with variance $-1/A_i$ (and covariance zero). Hence, we can set $J_{ii} = A_i$. The mean is $-[\mathbf{J}^{-1}\mathbf{h}]_i$. The exact marginals have mean $-[\mathbf{J}^{-1}\mathbf{h}]_i$ and co-variance $-[\mathbf{J}^{-1}]_{ij}$. The difference can be quite dramatic, e.g. in two dimensions for $\mathbf{J} = \begin{pmatrix} 1 & \epsilon \\ \epsilon & 1 \end{pmatrix}$, we get $\mathbf{J}^{-1} = \frac{1}{1-\epsilon^2} \begin{pmatrix} 1 & -\epsilon \\ -\epsilon & 1 \end{pmatrix}$. The variance estimates are $1/J_{ii} = 1$ for variational and $[\mathbf{J}^{-1}]_{ii} = 1/(1 - \epsilon^2)$ for the exact case. The latter diverges for completely correlated variable, $\epsilon \to 1$ illustrating that the variational covariance estimate breaks down when the interaction between the variables are strong.

A very important remark should be made at this point: although the covariance eq. (18) comes out correctly, the LR method *does not reproduce the exact two variable marginals*, i.e. the distribution eq. (9) plus the sum of the product of the one variable marginals is not a Gaussian distribution.

### 7.2 Mean and Variance of 1D Gaussian

A one dimensional Gaussian observation model $p(y|\mu, \beta) = \sqrt{\beta/2\pi} \exp(-\beta(x-\mu)^2/2)$, $\beta = 1/\sigma^2$ with a Gaussian prior over the mean and a Gamma prior over $\beta$ [7] serves as another example of where linear response—as opposed to variational—gives exact covariance estimates. The $N$ example likelihood can be rewritten as

$$p(\mathbf{y}|\mu, \beta) = \left( \frac{\beta}{2\pi} \right)^{\frac{N}{2}} \exp \left( -\frac{\beta}{2} N \hat{\sigma}^2 - \frac{\beta}{2} N (\mu - \overline{y})^2 \right) , \tag{19}$$

where $\overline{y}$ and $\hat{\sigma}^2 = \sum_i (y_i - \overline{y})^2/N$ are the empirical mean and variance. We immediately recognize $-\frac{\beta}{2} N (\mu - \overline{y})^2$ as the interaction term. Choosing non-informative priors—$p(\mu)$ flat and $p(\beta) \propto 1/\beta$—the variational distribution $q_\mu(\mu)$ becomes Gaussian with mean $\overline{y}$ and variance $1/N\langle \beta \rangle_q$ and $q_\beta(\beta)$ becomes a Gamma distribution $\Gamma(\beta|b, c) \propto \beta^{c-1} e^{-\beta/b}$, with parameters $c_q = N/2$ and $1/b_q = \frac{N}{2}(\hat{\sigma}^2 + \langle(\mu - \overline{y})^2\rangle_q)$. The mean and variance of Gamma distribution are given by $bc$ and $b^2 c$. Solving with respect to $\langle(\mu - \overline{y})^2\rangle_q$ and $\langle \beta \rangle_q$ give $1/b_q = \frac{N\hat{\sigma}^2}{2} \frac{N}{N-1}$. Exact inference gives $c_{\text{exact}} = (N-1)/2$ and $1/b_{\text{exact}} = \frac{N\hat{\sigma}^2}{2}$ [7]. A comparison shows that the mean $bc$ is the same in both cases whereas variational under-estimates the variance $b^2 c$. This is a quite generic property of the variational approach.

The LR correction to the covariance is easily derived from (13) setting $J_{12} = -N/2$ and $\phi_1(\beta) = \beta - \langle \beta \rangle_q$ and $\phi_2(\mu) = (\mu - \overline{y})^2 - \langle(\mu - \overline{y})^2\rangle_q$. This yields $J_{11} = -1/\langle \phi_1^2(\beta) \rangle = -1/b_q\langle \beta \rangle_q$. Using $\langle(\mu - \overline{y})^2\rangle_q = 1/N\langle \beta \rangle_q$ and $\langle(\mu - \overline{y})^4\rangle_q = 3\langle(\mu - \overline{y})^2\rangle_q^2$, we have $J_{22} = -1/\langle \phi_2^2(\mu) \rangle = -N^2\langle \beta \rangle_q^2/2$. Inverting the $2 \times 2$ matrix $\mathbf{J}$, we immediately get

$$\langle \phi_1^2 \rangle = \text{Var}(\beta) = -(\mathbf{J}^{-1})_{11} = b_q\langle \beta \rangle_q/(1 - b_q/2\langle \beta \rangle_q)$$

Inserting the result for $\langle \beta \rangle_q$, we find that this is in fact the correct result.

## 7.3 Variational Bayes Mixture of Multinomials

As a final example, we take a mixture model of practical interest and show that linear response corrections straightforwardly can be calculated. Here we consider the problem of modeling histogram data $y_{nj}$ consisting of $N$ histograms each with $D$ bins: $n = 1, \ldots, N$ and $j = 1, \ldots, D$. We can model this with a mixture of multinomials (Lars Kai Hansen 2003, in preparation):

$$p(\mathbf{y}_n|\boldsymbol{\pi}, \boldsymbol{\rho}) = \sum_{k=1}^{K} \pi_k \prod_{j=1}^{D} \rho_{kj}^{y_{nj}} , \qquad (20)$$

where $\pi_k$ is the probability of the $k$th mixture and $\rho_{kj}$ is the probability of observing the $j$th histogram given we are in the $k$th component, i.e. $\sum_k \pi_k = 1$ and $\sum_j \rho_{kj} = 1$. Eventually in the variational Bayes treatment we will introduce Dirichlet priors for the variables. But the general linear response expression is independent of this. To rewrite the model such that it is suitable for a variational treatment—i.e. in a product form—we introduce hidden (Potts) variables $\mathbf{x}_n = \{x_{nk}\}$, $x_{nk} = \{0, 1\}$ and $\sum_k x_{nk} = 1$ and write the joint probability of observed and hidden variables as:

$$p(\mathbf{y}_n, \mathbf{x}_n|\boldsymbol{\pi}, \boldsymbol{\rho}) = \prod_{k=1}^{K} \left( \pi_k \prod_{j=1}^{D} \rho_{kj}^{y_{nj}} \right)^{x_{nk}} . \qquad (21)$$

Summing over all possible $\mathbf{x}_n$ vectors, we recover the original observation model.

We can now identify the interaction terms in $\sum_n \ln p(\mathbf{y}_n, \mathbf{x}_n, \boldsymbol{\pi}, \boldsymbol{\rho})$ as $x_{nk} \ln \pi_k$ and $y_{nj} x_{nk} \ln \rho_{kj}$. Generalizing eq. (8) to sets of variables, we can compute the following correlations $C(\boldsymbol{\pi}, \boldsymbol{\pi}')$, $C(\boldsymbol{\pi}, \boldsymbol{\rho}')$ and $C(\boldsymbol{\rho}_k, \boldsymbol{\rho}'_{k'})$. To get the explicit solution we need to write the coupling matrix for the problem and add diagonal terms and invert. Normally, the complexity will be order cubed in the number of parameters. However, it turns out that the two variable marginal distributions involving the hidden variables—the number of which scales with the number of examples—can be eliminated analytically. The computation of correlation are thus only cubic in the number of parameters, $K + K * D$, making computation of correlations attractive even for mixture models.

The symmetric coupling matrix for this problem can be written as

$$\mathbf{J} = \begin{pmatrix} \mathbf{J}_{\mathbf{xx}} & \mathbf{J}_{\mathbf{x}\boldsymbol{\pi}} & \mathbf{J}_{\mathbf{x}\boldsymbol{\rho}} \\ \mathbf{J}_{\boldsymbol{\pi}\mathbf{x}} & \mathbf{J}_{\boldsymbol{\pi}\boldsymbol{\pi}} & \mathbf{J}_{\boldsymbol{\pi}\boldsymbol{\rho}} \\ \mathbf{J}_{\boldsymbol{\rho}\mathbf{x}} & \mathbf{J}_{\boldsymbol{\rho}\boldsymbol{\pi}} & \mathbf{J}_{\boldsymbol{\rho}\boldsymbol{\rho}} \end{pmatrix} \text{ with } \mathbf{J}_{\boldsymbol{\rho}\mathbf{x}} = \begin{pmatrix} \mathbf{J}_{\boldsymbol{\rho}_1 \mathbf{x}_1} & \cdots & \mathbf{J}_{\boldsymbol{\rho}_1 \mathbf{x}_N} \\ \vdots & & \vdots \\ \mathbf{J}_{\boldsymbol{\rho}_K \mathbf{x}_1} & \cdots & \mathbf{J}_{\boldsymbol{\rho}_K \mathbf{x}_N} \end{pmatrix} , \qquad (22)$$

where for simplicity the log on $\pi$ and $\rho$ are omitted and $(\mathbf{J}_{\boldsymbol{\rho}_k \mathbf{x}_n})_{jk} = y_{nj}$. The other non-zero sub-matrix is: $\mathbf{J}_{\boldsymbol{\pi}\mathbf{x}} = [\mathbf{J}_{\boldsymbol{\pi}\mathbf{x}_1} \cdots \mathbf{J}_{\boldsymbol{\pi}\mathbf{x}_N}]$ with $(\mathbf{J}_{\boldsymbol{\pi}\mathbf{x}_n})_{kk'} = \delta_{k,k'}$. To get the covariance $\mathbf{V}$ we introduce diagonal elements into $\mathbf{J}$ (which are all tractable in $\langle \ldots \rangle = \langle \ldots \rangle_q$):

$$-(\mathbf{J}_{\mathbf{x}_n \mathbf{x}_n}^{-1})_{kk'} = \langle x_{nk} x_{nk'} \rangle - \langle x_{nk} \rangle \langle x_{nk'} \rangle = \delta_{kk'} \langle x_{nk} \rangle - \langle x_{nk} \rangle \langle x_{nk'} \rangle \qquad (23)$$

$$-(\mathbf{J}_{\boldsymbol{\pi}\boldsymbol{\pi}}^{-1})_{kk'} = \langle \ln \pi_k \ln \pi_{k'} \rangle - \langle \ln \pi_k \rangle \langle \ln \pi_{k'} \rangle \qquad (24)$$

$$-(\mathbf{J}_{\boldsymbol{\rho}_k \boldsymbol{\rho}_k}^{-1})_{jj'} = \langle \ln \rho_{kj} \ln \rho_{kj'} \rangle - \langle \ln \rho_{kj} \rangle \langle \ln \rho_{kj'} \rangle \qquad (25)$$

and invert: $\mathbf{V} = -\mathbf{J}^{-1}$.

Using inversion by partitioning and the Woodbury formula we find the following simple formula

$$\mathbf{V}_{\boldsymbol{\pi}\boldsymbol{\pi}} = \left( \hat{\mathbf{J}}_{\boldsymbol{\pi}\boldsymbol{\pi}} - \mathbf{J}_{\boldsymbol{\pi}\boldsymbol{\pi}} - \hat{\mathbf{J}}_{\boldsymbol{\pi}\boldsymbol{\rho}} \left( \hat{\mathbf{J}}_{\boldsymbol{\rho}\boldsymbol{\rho}} - \mathbf{J}_{\boldsymbol{\rho}\boldsymbol{\rho}} \right)^{-1} \hat{\mathbf{J}}_{\boldsymbol{\rho}\boldsymbol{\pi}} \right)^{-1} , \qquad (26)$$

where we have introduced the 'indirect' couplings $\hat{\mathbf{J}}_{\boldsymbol{\pi}\boldsymbol{\pi}} = \mathbf{J}_{\boldsymbol{\pi}\mathbf{x}} \mathbf{J}_{\mathbf{xx}}^{-1} \mathbf{J}_{\mathbf{x}\boldsymbol{\pi}}$ and $\hat{\mathbf{J}}_{\boldsymbol{\pi}\boldsymbol{\rho}} = \mathbf{J}_{\boldsymbol{\pi}\mathbf{x}} \mathbf{J}_{\mathbf{xx}}^{-1} \mathbf{J}_{\mathbf{x}\boldsymbol{\rho}}$. Similar formulas can be obtained for $\mathbf{V}_{\boldsymbol{\pi}\boldsymbol{\rho}}$ and $\mathbf{V}_{\boldsymbol{\rho}\boldsymbol{\rho}}$.

## 8 Conclusion and Outlook

In this paper we have shown that it is possible to extend linear response to completely general variational distributions and solve the linear response equations explicitly. We have given three applications that show 1. that linear response provides approximations of increased quality for two-variable marginals and 2. linear response is practical for variational Bayes models. Together this suggests that building linear response into variational Bayes software such as VIBES [5, 6] would be useful.

Welling and Teh [12, 13] have, as mentioned in the introduction, shown how to apply the general linear response methods to the Bethe approximation. However, the usefulness of linear response even goes beyond this: if we can come up with a better tractable approximation to the marginal distribution $q(s_i)$ with some free parameters, we can tune these parameters by requiring consistency between $q(s_i)$ and the linear response estimate of the diagonal of the two-variable marginals eq. (8):

$$C_{ii}(s_i, s'_i) = \delta(s_i - s'_i)q(s_i) - q(s_i)q(s'_i) \ . \tag{27}$$

This design principle can be generalized to models that give non-trivial estimates of two-variable marginals such as Bethe. It might not be possible to match the entire distribution for a tractable choice of $q(s_i)$. In that case it is possibly to only require consistency for some statistics. The adaptive TAP approach [11]—so far only studied for quadratic interactions—can be viewed in this way. Generalizing this idea to general potentials, general mean field approximations, deriving the corresponding marginal likelihoods and deriving guaranteed convergent algorithms for the approximations are under current investigation.

## Footnotes

[1]The functional derivative is defined by $\frac{\delta a_j(s_j)}{\delta a_i(s_i)} = \delta_{ij}\delta(s_i - s_j)$ and the chain rule.

## References

[1] M. Opper and D. Saad, *Advanced Mean Field Methods: Theory and Practice*, MIT Press, 2001.

[2] H. Attias, "A variational Bayesian framework for graphical models," in *Advances in Neural Information Processing Systems 12*, T. Leen et al., Ed. 2000, MIT Press, Cambridge.

[3] J. W. Miskin and D. J. C. MacKay, "Ensemble learning for blind image separation and deconvolution," in *Advances in Independent Component Analysis*, M Girolami, Ed. 2000, Springer-Verlag Scientific Publishers.

[4] Z. Ghahramani and M. J. Beal, "Propagation algorithms for variational Bayesian learning," in *Advances in Neural Information Processing Systems 13*. 2001, pp. 507–513, MIT Press, Cambridge.

[5] C. M. Bishop, D. Spiegelhalter, and J. Winn, "VIBES: A variational inference engine for Bayesian networks," in *Advances in Neural Information Processing Systems 15*, 2002.

[6] C. M. Bishop and J. Winn, "Structured variational distributions in VIBES," in *Artificial Intelligence and Statistics, Key West, Florida*, 2003.

[7] D. J. C. MacKay, *Information Theory, Inference, and Learning Algorithms*, Cambridge University Press, 2003.

[8] G. Parisi, *Statistical Field Theory*, Addison-Wesley, 1988.

[9] H.J. Kappen and F.B. Rodríguez, "Efficient learning in Boltzmann machines using ' linear response theory," *Neural Computation*, vol. 10, pp. 1137–1156, 1998.

[10] P. A.d.F.R. Hojen-Sorensen, O. Winther, and L. K. Hansen, "Mean field approaches to independent component analysis," *Neural Computation*, vol. 14, pp. 889–918, 2002.

[11] M. Opper and O. Winther, "Adaptive and self-averaging Thouless-Anderson-Palmer mean field theory for probabilistic modeling," *Physical Review E*, vol. 64, pp. 056131, 2001.

[12] M. Welling and Y. W. Teh, "Linear response algorithms for approximate inference," *Artificial Intelligence Journal*, 2003.

[13] M. Welling and Y. W. Teh, "Propagation rules for linear response estimates of joint pairwise probabilities," *preprint*, 2003.
